# LEARNING BY CHOICE
# OF INTERNAL REPRESENTATIONS

Tal Grossman, Ronny Meir and Eytan Domany
Department of Electronics, Weizmann Institute of Science
Rehovot 76100 Israel

## ABSTRACT

We introduce a learning algorithm for multilayer neural networks composed of binary linear threshold elements. Whereas existing algorithms reduce the learning process to minimizing a cost function over the *weights*, our method treats the *internal representations* as the fundamental entities to be determined. Once a correct set of internal representations is arrived at, the weights are found by the local and biologically plausible Perceptron Learning Rule (PLR). We tested our learning algorithm on four problems: adjacency, symmetry, parity and combined symmetry-parity.

## I. INTRODUCTION

Consider a network of binary linear threshold elements $i$, whose state $S_i = \pm 1$ is determined according to the rule

$$S_i = sign(\sum_j w_{ij} S_j + \theta_i) \quad . \tag{1}$$

Here $w_{ij}$ is the (unidirectional) weight assigned to the connection from unit $j$ to $i$; $\theta_i$ is a local bias. We focus our attention on feed-forward networks in which $N$ units of the input layer determine the states of $H$ units of a hidden layer; these, in turn, feed one or more output elements.

For a typical *A vs B classification* task such a network needs a single output, with $S^{out} = +1$ (or -1) when the input layer is set in a state that belongs to category $A$ (or $B$) of input space. The basic problem of learning is to find an algorithm, that produces weights which enable the network to perform this task. In the absence of hidden units learning can be accomplished by the PLR [Rosenblatt 1962], which we now briefly describe. Consider $j = 1, ..., N$ source units and a single target unit $i$. When the source units are set in any one of $\mu = 1, ..M$ patterns, i.e. $S_j = \xi_j^\mu$, we require that the target unit (determined using (1)) takes preassigned values $\xi_i^\mu$. Learning takes place in the course of a training session. Starting from any arbitrary initial guess for the weights, an input $\nu$ is presented, resulting in the output taking some value $S_i^\nu$. Now modify every weight according to the rule

$$w_{ij} \rightarrow w_{ij} + \eta(1 - S_i^\nu \xi_i^\nu)\xi_i^\nu \xi_j^\nu \quad , \tag{2}$$

where $\eta > 0$ is a parameter ($\xi_j^\nu = 1$ is used to modify the bias $\theta$). Another input pattern is presented, and so on, until all inputs draw the correct output. The *Perceptron convergence theorem* states [Rosenblatt 1962, Minsky and Papert 1969] that the PLR will find a solution (if one exists), in a finite number of steps. However, of the $2^{2^N}$ possible partitions of input space only a small subset (less than $2^{N^2}/N!$) is linearly separable [Lewis and Coates 1967], and hence soluble by single-layer perceptrons. To get around this, hidden units are added. Once a single hidden layer (with a large enough number of units) is inserted beween input and output, every classification problem has a solution. But for such architectures the PLR cannot be implemented; when the network errs, it is not clear which connection is to blame for the error, and what corrective action is to be taken.

Back-propagation [Rumelhart et al 1986] circumvents this "credit-assignment" problem by dealing only with networks of continuous valued units, whose response function is also continuous (sigmoid). "Learning" consists of a gradient-descent type minimization of a cost function that measure the deviation of actual outputs from the required ones, in the space of weights $w_{ij}, \theta_i$. A new version of BP, "back propagation of desired states", which bears some similarity to our algorithm, has recently been introduced [Plaut 1987]. See also le Cun [1985] and Widrow and Winter [1988] for related methods.

Our algorithm views the internal representations associated with various inputs as the basic independent variables of the learning process. This is a conceptually plausible assumption; in the course of learning a biological or artificial system should form maps and representations of the external world. Once such representations are formed, the weights can be found by simple and local Hebbian learning rules such as the PLR. Hence the problem of learning becomes one of *searching for proper internal representations*, rather than one of minimization. Failure of the PLR to converge to a solution is used as an indication that the current guess of internal representations needs to be modified.

## II. THE ALGORITHM

If we know the internal representations (e.g. the states taken by the hidden layer when patterns from the training set are presented), the weights can be found by the PLR. This way the problem of learning becomes one of choosing proper internal representations, rather than of minimizing a cost function by varying the values of weights. To demonstrate our approache, consider the classification problem with output values, $S^{out,\mu} = \xi^{out,\mu}$, required in response to $\mu = 1, ..., M$ input patterns. If a solution is found, it first maps each input onto an internal representation generated on the hidden layer, which, in turn, produces the correct output. Now imagine that we are not supplied with the weights that solve the problem; however the correct internal representations are revealed. That is, we are given a *table* with $M$ rows, one for each input. Every row has $H$ bits $\xi_i^{h,\mu}$, for $i = 1, ..H$, specifying the state of the hidden layer obtained in response to input pattern $\mu$. One can now view each hidden-layer cell $i$ as the target cell of the PLR, with the $N$ inputs viewed as source. Given sufficient time, the PLR will converge to a set

of weights $W_{i,j}$, connecting input unit $j$ to hidden unit $i$, so that indeed the input-output association that appears in column $i$ of our table will be realized. In a similar fashion, the PLR will yield a set of weights $w_i$, in a learning process that uses the hidden layer as source and the output unit as target. Thus, in order to solve the problem of learning, all one needs is a search procedure in the space of possible internal representations, for a table that can be used to generate a solution. Updating of weights can be done in parallel for the two layers, using the current table of internal representations. In the present algorithm, however, the process is broken up into four distinct stages:

1. **SETINREP**: Generate a table of internal representations $\{\xi_i^{h,\nu}\}$ by presenting each input pattern from the training set and calculating the state on the hidden layer,using Eq.(1a), with the existing couplings $W_{ij}$ and $\theta_i$.

2. **LEARN23**: The hidden layer cells are used as source, and the output as the target unit of the PLR. The current table of internal representations is used as the training set; the PLR tries to find appropriate weights $w_i$ and $\theta$ to obtain the desired outputs. If a solution is found, the problem has been solved. Otherwise stop after $I_{23}$ learning sweeps, and keep the current weights, to use in INREP.

3. **INREP**: Generate a new table of internal representations, which, when used in (1b), yields the correct outputs. This is done by presenting the table sequentially, row by row, to the hidden layer. If for row $\nu$ the wrong output is obtained, the internal representation $\xi^{h,\nu}$ is changed. Having the wrong output means that the "field" produced by the hidden layer on the output unit, $h^{out,\nu} = \sum_j w_j \xi_j^{h,\nu}$ is either too large or too small. We then randomly pick a site $j$ of the hidden layer, and try to flip the sign of $\xi_j^{h,\nu}$; if $h^{out,\nu}$ changes in the right direction, we replace the entry of our table, i.e.

$$\text{if} \quad w_j \xi_j^{h,\nu} \xi^{out,\nu} < 0 \quad \text{then} \quad \xi_j^{h,\nu} \to -\xi_j^{h,\nu}.$$

We keep picking sites and changing the internal representation of pattern $\nu$ until the correct output is generated. We always generate the correct output this way, provided $\sum_j |w_j| > |\theta^{out}|$ (as is the case for our learning process in LEARN23). This procedure ends with a modified table which is our next guess of internal representations.

4. **LEARN12**: Apply the PLR with the first layer serving as source, treating every hidden layer site separately as target. Actually, when an input from the training set is presented to the first layer, we first check whether the correct result is produced on the *output* unit of the network. If we get wrong overall output, we use the PLR for every hidden unit $i$, modifying weights incident on $i$ according to (2), using column $i$ of the table as the desired states of this unit. If input $\nu$ does yield the correct output, we insert the current state of the hidden layer as the internal representation associated with pattern $\nu$, and no learning steps are taken. We sweep in this manner the training set, modifying weights $W_{ij}$, (between input and hidden layer), hidden-layer thresholds $\theta_i$, and, as explained above, internal

representations. If the network has achieved error-free performance for the entire training set, learning is completed. If no solution has been found after $I_{12}$ sweeps of the training set, we abort the PLR stage, keep the present values of $w_{ij}, \theta_i$, and start SETINREP again.

This is a fairly complete account of our procedure (see also Grossman et al [1988]). There are a few details that need to be added.

a) The "impatience" parameters: $I_{12}$ and $I_{23}$, which are rather arbitrary, are introduced to guarantee that the PLR stage is aborted if no solution is found. This is necessary since it is not clear that a solution exists for the weights, given the current table of internal representations. Thus, if the PLR stage does not converge within the time limit specified, a new table of internal representations is formed. The parameters have to be large enough to allow the PLR to find a solution (if one exists) with sufficiently high probability. On the other hand, too large values are wasteful, since they force the algorithm to execute a long search even when no solution exists. Therefore the best values of the impatience parameters can be determined by optimizing the performance of the network; our experience indicates, however, that once a "reasonable" range of values is found, performance is fairly insensitive to the precise choice.

b) Integer weights: In the PLR correction step, as given by Eq.2, the size of $\Delta W$ is constant. Therefore, when using binary units, it can be scaled to unity (by setting $\eta = 0.5$) and one can use integer $W_{i,j}$'s without any loss of generality.

c) Optimization: The algorithm described uses several parameters, which should be optimized to get the best performance. These parameters are: $I_{12}$ and $I_{23}$ - see section (a) above; $I_{max}$ - time limit, i.e. an upper bound to the total number of training sweeps; and the PLR training parameters - i.e the increment of the weights and thresholds during the PLR stage. In the PLR we used values of $\eta \simeq 0.1$ [see Eq. (2) ] for the weights, and $\eta \simeq 0.05$ for thresholds, whereas the initial (random) values of all weights were taken from the interval (-0.5,0.5), and thresholds from (-0.05,0.05). In the integer weights program, described above, these parameters are not used.

d) Treating Multiple Outputs: In the version of *inrep* described above, we keep flipping the internal representations until we find one that yields the correct output, i.e. zero error for the given pattern. This is not always possible when using more than one output unit. Instead, we can allow only for a pre-specified number of attempted flips, $I_{in}$, and go on to the next pattern even if vanishing error was not achieved. In this modified version we also introduce a slightly different, and less "restrictive" criterion for accepting or rejecting a flip. Having chosen (at random) a hidden unit $i$, we check the effect of flipping the sign of $\xi_i^{h,\nu}$ on the total output error, i.e. the number of wrong bits (and *not* on the output field, as described above). If the output error is not increased, the flip is accepted and the table of internal representations is changed accordingly.

This modified algorithm is applicable for multiple-output networks. Results of preliminary experiments with this version are presented in the next section.

## III. PERFORMANCE OF THE ALGORITHM

The "time" parameter that we use for measuring performance is the number of sweeps through the training set of M patterns needed in order to find the solution. Namely, how many times each pattern was presented to the network. In each cycle of the algorithm there are $I_{12} + I_{23}$ such sweeps. For each problem, and each parameter choice, an ensemble of many independent runs, each starting with a different random choice of initial weights, is created. In general, when applying a learning algorithm to a given problem, there are cases in which the algorithm fails to find a solution within the specified time limit (e.g. when BP get stuck in a local minimum), and it is impossible to calculate the ensemble average of learning times. Therefore we calculate, as a performance measure, either the median number of sweeps, $t_m$, or the "inverse average rate", $\tau$, as defined in Tesauro and Janssen [1988].

The first problem we studied is *contiguity*: the system has to determine whether the number of clumps (i.e. contiguous blocks) of +1's in the input is, say, equal to 2 or 3. This is called [Denker et al 1987] the "2 versus 3" clumps predicate. We used, as our training set, all inputs that have 2 or 3 clumps, with learning cycles parametrized by $I_{12} = 20$ and $I_{23} = 5$. Keeping $N = 6$ fixed, we varied $H$; 500 cases were used for each data point of Fig.1.

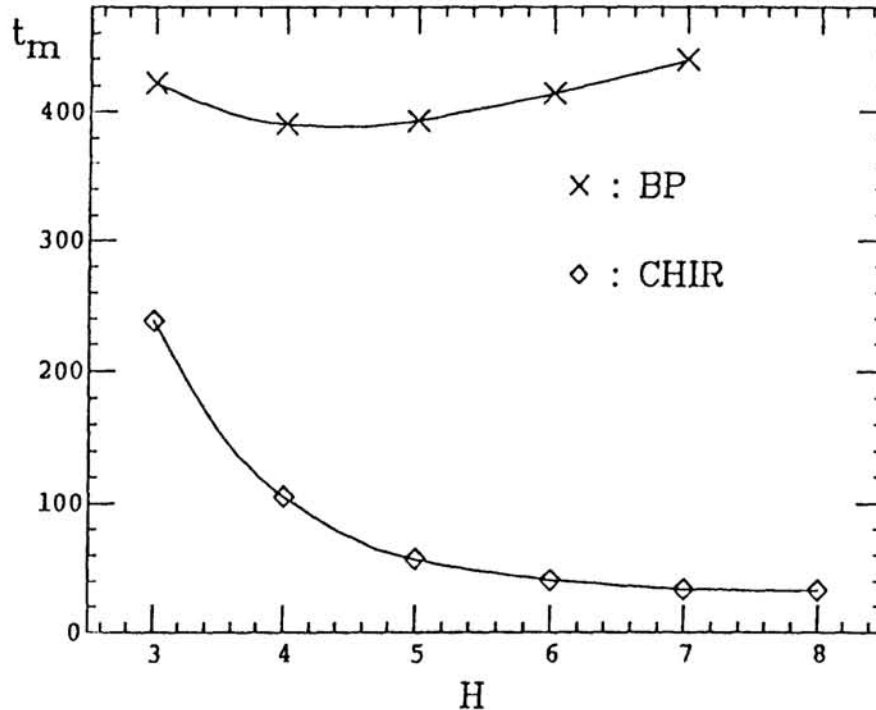

**Figure 1.**    Median number of sweeps $t_m$, needed to train a network of $N = 6$ input units, over an exhaustive training set, to solve the " 2 vs 3" clumps predicate, plotted against the number of hidden units $H$. Results for back-propagation [Denker et al 1987] ($\times$) and this work ($\Diamond$) are shown.

In the next problem, *symmetry*, one requires $S^{out} = 1$ for reflection-symmetric inputs and $-1$ otherwise. This can be solved with $H \geq 2$ hidden units. Fig. 2 presents, for $H = 2$, the median number of exhaustive training sweeps needed to solve the problem, vs input size $N$. At each point 500 cases were run, with $I_{12} = 10$ and $I_{23} = 5$. We always found a solution in less than 200 cycles.

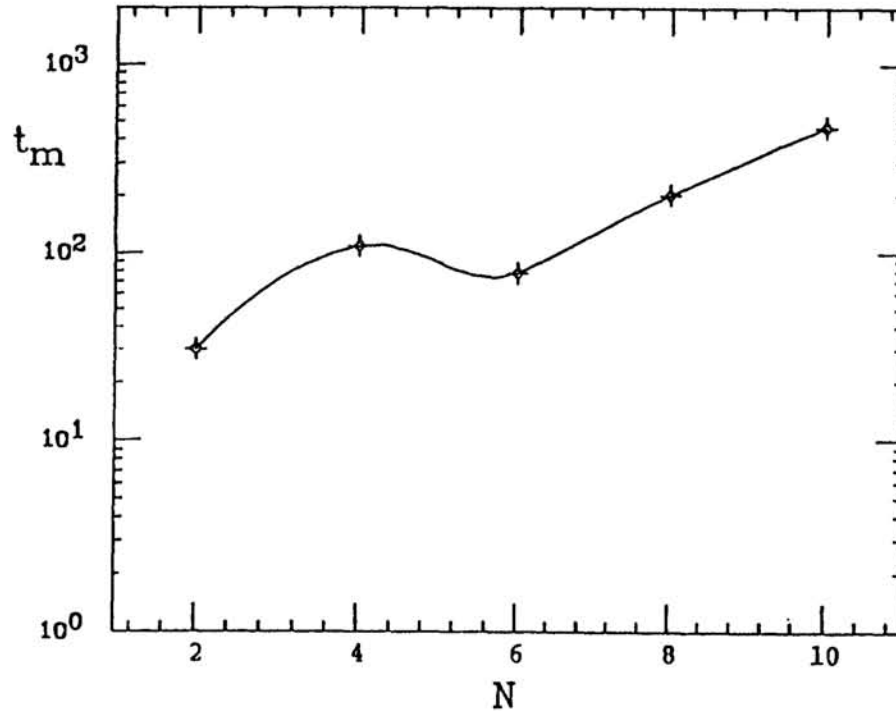

**Figure 2.**     Median number of sweeps $t_m$, needed to train networks on symmetry (with $H = 2$).

In the *Parity* problem one requires $S^{out} = 1$ for an even number of $+1$ bits in the input, and $-1$ otherwise. In order to compare performance of our algorithm to that of BP, we studied the Parity problem, using networks with an architecture of $N : 2N : 1$, as chosen by Tesauro and Janssen [1988].

We used the integer version of our algorithm, briefly described above. In this version of the algorithm the weights and thresholds are integers, and the increment size, for both thresholds and weights, is unity. As an initial condition, we chose them to be $+1$ or $-1$ randomly. In the simulation of this version, all possible input patterns were presented sequentially in a fixed order (within the perceptron learning sweeps). The results are presented in Table 1. For all choices of the parameters ( $I_{12}$, $I_{23}$ ), that are mentioned in the table, our success rate was 100%. Namely, the algorithm didn't fail even once to find a solution in less than the maximal number of training cycles $I_{max}$ specified in the table. Results for BP, $\tau(BP)$ (from Tesauro and Janssen 1988) are also given in the table. Note that BP does get caught in local minima, but the percentage of such occurrences was not reported.

For testing the multiple output version of the algorithm we used the combined parity and symmetry problem; the network has two output units, both connected to all hidden units. The first output unit performs the parity predicate on the input, and the second performs the symmetry predicate. The network architecture was N:2N:2 and the results for N=4..7 are given in Table 2. The choice of parameters is also given in that table.

| N | $(I_{12}, I_{23})$ | $I_{max}$ | $t_m$ | $\tau(CHIR)$ | $\tau(BP)$ |
|---|---|---|---|---|---|
| 3 | (8,4) | 10 | 3 | 3 | 33 |
| 4 | (9,3)(6,6) | 20 | 4 | 4 | 75 |
| 5 | (12,4)(9,6) | 40 | 8 | 6 | 130 |
| 6 | (12,4)(10,5) | 120 | 19 | 9 | 310 |
| 7 | (12,4)(15,5) | 240 | 290 | 30 | 800 |
| 8 | (20,10) | 900 | 2900 | 150 | 2000 |
| 9 | (20,10) | 900 | 2400 | 1300 | - |

Table 1. Parity with N:2N:1 architecture.

| N | $I_{12}$ | $I_{23}$ | $I_{in}$ | $I_{max}$ | $t_m$ | $\tau$ |
|---|---|---|---|---|---|---|
| 4 | 12 | 8 | 7 | 40 | 50 | 33 |
| 5 | 14 | 7 | 7 | 400 | 900 | 350 |
| 6 | 18 | 9 | 7 | 900 | 5250 | 925 |
| 7 | 40 | 20 | 7 | 900 | 6000 | 2640 |

Table 2. Parity and Symmetry with N:2N:2 architecture.

## IV. DISCUSSION

We have presented a learning algorithm for two-layer perceptrons, that searches for internal representations of the training set, and determines the weights by the local, Hebbian perceptron learning rule. Learning by choice of internal representation may turn out to be most useful in situations where the "teacher" has some information about the desired internal representations.

We demonstrated that our algorithm works well on four typical problems, and studied the manner in which training time varies with network size. Comparisons with back-propagation were also made. It should be noted that a training sweep involves much less computations than that of back-propagation. We also presented a generalization of the algorithm to networks with multiple outputs, and found that it functions well on various problems of the same kind as discussed above. It appears that the modification needed to deal with multiple outputs also enables us to solve the learning problem for network architectures with more than one hidden layer.

At this point we can offer only very limited discussion of the interesting question - why does our algorithm work at all? That is, how come it finds correct internal representations (e.g. "tables") while these constitute only a small fraction of the total possible number $(2^{H2^N})$? The main reason is that our procedure actually does not search this entire space of tables. This large space contains a small subspace, $T$, of "target tables", i.e. those that can be obtained, for all possible choices of $w_{ij}^l$ and $\theta_i$, by rule (1), in response to presentation of the input patterns. Another small subspace $S$, is that of the tables that can potentially produce the required output. Solutions of the learning problem constitute the space $T \cap S$. Our algorithm iterates between $T$ and $S$, executing also a "walk" (induced by the modification of the weights due to the PLR) within each.

An appealing feature of our algorithm is that it can be implemented in a manner that uses only integer-valued weights and thresholds. This discreteness makes the analysis of the behavior of the network much easier, since we know the exact number of bits used by the system in constructing its solution, and do not have to worry about round-off errors. From a technological point of view, for hardware implementation it may also be more feasible to work with integer weights.

We are extending this work in various directions. The present method needs, in the learning stage, $MH$ bits of memory: internal representations of all $M$ training patterns are stored. This feature is biologically implausible and may be technologically limiting; we are developing a method that does not require such memory. Other directions of current study include extensions to networks with continuous variables, and to networks with feed-back.

## References

Denker J., Schwartz D., Wittner B., Solla S., Hopfield J.J., Howard R. and Jackel L. 1987, *Complex Systems* 1, 877-922

Grossman T., Meir R. and Domany E. 1988, *Complex Systems* in press.

Hebb D.O. 1949, *The organization of Behavior*, J. Wiley, N.Y

Le Cun Y. 1985, *Proc. Cognitiva* 85, 593

Lewis P.M. and Coates C.L. 1967, *Threshold Logic*. (Wiley, New York)

Minsky M. and Papert S. 1988, *Perceptrons*. (MIT, Cambridge).

Plaut D.C., Nowlan S.J. and Hinton G.E. 1987, Tech. Report **CMU-CS-86-126**

Rosenblatt F. *Principles of neurodynamics*. (Spartan, New York, 1962)

Rumelhart D.E., Hinton G.E. and Williams R.J. 1986, *Nature* 323, 533-536

Tesauro G. and Janssen H. 1988, *Complex Systems* 2, 39

Widrow B. and Winter R. 1988, *Computer* 21, No.3, 25
